# Cross Species Expression Analysis using a Dirichlet Process Mixture Model with Latent Matchings

**Hai-Son Le**
Machine Learning Department
Carnegie Mellon University
Pittsburgh, PA, USA
hple@cs.cmu.edu

**Ziv Bar-Joseph**
Machine Learning Department
Carnegie Mellon University
Pittsburgh, PA, USA
zivbj@cs.cmu.edu

## Abstract

Recent studies compare gene expression data across species to identify core and species specific genes in biological systems. To perform such comparisons researchers need to match genes across species. This is a challenging task since the correct matches (orthologs) are not known for most genes. Previous work in this area used deterministic matchings or reduced multidimensional expression data to binary representation. Here we develop a new method that can utilize soft matches (given as priors) to infer both, unique and similar expression patterns across species and a matching for the genes in both species. Our method uses a Dirichlet process mixture model which includes a latent data matching variable. We present learning and inference algorithms based on variational methods for this model. Applying our method to immune response data we show that it can accurately identify common and unique response patterns by improving the matchings between human and mouse genes.

## 1 Introduction

Researchers have been increasingly relying on cross species analysis to understand how biological systems operate. Sequence based methods have been successfully applied to identify and characterize coding and functional non coding regions in multiple species [1]. However, sequence information is static and thus provides only partial view of cellular activity. More recent studies attempt to integrate sequence and gene expression data from multiple species [2, 3, 4]. Unlike sequence, expression levels are dynamic and differ across time and conditions. By combining expression and sequence data researchers were able to identify both "core" and "divergent" genes. "Core" genes are similarly expressed across species and are useful for constructing models of conserved systems, for example the cell cycle [2]. "Divergent" genes are similar in sequence but differ in expression across species. These are useful for identifying species specific responses, for example why some pathogens are resistant to drugs while others are not [3].

While useful, cross species analysis of expression data is challenging. In addition to the regular issues with expression data (noise, missing values, etc.) when comparing expression levels across species researchers need to match genes across species. For most genes the correct match in another species (known as ortholog) is not known. A number of methods have been suggested to solve the matching problem. The first set of methods is based on a one to one deterministic assignment by relying on top sequence matches. Such an assignment can be used to concatenate the expression vectors for matched genes across species and then cluster the resulting vectors. For example, Stuart et al. [5] constructed "metagenes" consisting of top sequence matches from four species. These were used to cluster the data from multiple species to identify conserved and divergent patterns. Bergmann et al. [6] defined one of the species (species A) as a reference and first clustered genes in A. They then used matched genes in the second species (B) as starting points for clustering

genes in B. When the clustering algorithm converges in B, genes that remain in the cluster are considered "core" whereas genes that are removed are "divergent". Quon et al. [4] used a mixture of Gaussians model, which takes as input the expression data of orthologous genes and a phylogenetic tree connecting the species, to reconstruct the expression profiles as well as detecting divergent links in the phylogeny. The second set of methods allowed for soft matches but was either limited to analyzing binary or discrete data with very few labels. For example, Lu et al. combined experiments from multiple species by using Markov Random Fields [7] and Gaussian Random Fields [8] in which edges represent sequence similarity and potential functions constrain similar genes across species to have a similar expression pattern.

While both approaches led to successful applications, they suffer from drawbacks that limit their use in practice. In many cases the top sequence match is not the correct ortholog and a deterministic assignment may lead to wrong conclusions about the conservation of genes. Methods that have used soft assignments were limited to summarization of the data (up or down regulated) and could not utilize more complex profiles. Here we present a new method that uses soft assignments to allow comparison and clustering across species of arbitrary expression data without requiring prior knowledge on the phylogeny. Our method takes as input expression datasets in two species and a prior on matches between homologous genes in these species (derived from sequence data). The method simultaneously clusters the expression values for both species while computing a posterior for the assignment of orthologs for genes. We use Dirichlet Process model to automatically detect the number of clusters.

We have tested our method on simulated and immune response data. In both cases the algorithm was able to find correct matches and to improve upon methods that used a deterministic assignment. While the method was developed for, and applied to, biological data, it is general and can be used to address other problems including matchings of captions to images (see Section 5).

## 2 Problem definition

In this section, we first describe in details the cross species analysis problem for gene expression data. Next, we formalize this as a general clustering and matching problem for cases in which the matches are not known in advance.

Using microarrays or new sequencing techniques researchers can monitor the expression levels of genes under certain conditions or at specific time points. For each such measurement we obtain a vector whose elements are the expression values for all genes (there are usually thousands of entries in each vector). We assume that the input consists of microarray experiments from two species and each species has a different set of genes. While the exact matches between genes in both species are not known for most genes, we have a prior for gene pairs (one from each species) which is derived from sequence data [9]. Our goal is to simultaneously cluster the genes in both species. Such clustering can identify coherent and divergent responses between the species. In addition, we would like to infer for each gene in one species whether there exists a homolog that is similarly expressed in the other species and if so, who.

The problem can also be formalized more generally in the following way. Denote by $\mathbf{x} = [x_1, x_2, \ldots, x_{n_x}]$ and $\mathbf{y} = [y_1, y_2, \ldots, y_{n_y}]$ the datasets of samples from two different experiment settings, where $x_i \in \Re^{p_x}$ and $y_j \in \Re^{p_y}$. In addition, let $\mathcal{M}$ be a sparse non-negative $n_x \times n_y$ matrix that encodes prior information regarding the matching of samples in $\mathbf{x}$ and $\mathbf{y}$. We define the match probability between $x_i$ and $y_j$ as follows:

$$p(x_i \text{ and } y_j \text{ are matched}) = \frac{\mathcal{M}(i,j)}{N_i} = \pi_{i,j} \qquad p(x_i \text{ is not matched}) = \frac{1}{N_i} = \pi_{i,0} \qquad (1)$$

where $N_i = 1 + \sum_{j=1}^{n_y} \mathcal{M}(i,j)$. $\pi_{i,0}$ is the prior probability that $x_i$ is not matched to any element in $Y$. We use $\boldsymbol{\pi_i}$ to denote the vector $(\pi_{i,0}, \ldots, \pi_{i,n_y})$. Finally, let $m_i \in \{0, 1, \ldots, n_y\}$ be the latent matching variable. If $m_i = 1$ we say that $x_i$ is matched to $y_{m_i}$. If $m_i = 0$ for we say that $x_i$ has no match in $\mathbf{y}$. Our goal is to infer both, the latent variables $m_j$'s and cluster membership for pairs of samples $(x_i, y_{m_i})$'s. The following notations are used in the rest of the paper. Lowercase normal font, e.g $x$, is used for a single variable and lowercase bold font, e.g $\mathbf{x}$, is used for vectors. Uppercase bold roman letters, such as $\mathbf{M}$, denote matrices. Uppercase letters, e.g $X$, are used to represent random variables and $\mathrm{E}[X]$ represents the expectation of a random variable $X$.

# 3 Model

Model selection is an important problem when analyzing real world data. Many clustering algorithms, including Gaussian mixture models, require as an input the number of clusters. In addition to domain knowledge, this model selection question can be addressed using cross validation. Bayesian nonparametric methods provide an alternative solution allowing the complexity of the model to grow based on the amount of available data. Under-fitting is addressed by the fact that the model allows for unbounded complexity while over-fitting is mitigated by the Bayesian assumption. We use this approach to develop a nonparametric model for clustering and matching cross species expression data. Our model, termed Dirichlet Process Mixture Model with Latent Matchings (DPMMLM) extends the popular Dirichlet Process Mixture Model to cases where priors are provided to matchings between vectors to be clustered.

## 3.1 Dirichlet Process

Let $G_0$ a probability measure on a measurable space. We write $G \sim DP(\alpha, G_0)$ if G is a random probability measure drawn from a Dirichlet process (DP). The existence of the Dirichlet process was first proven by [10]. Furthermore, measures of $G$ are discrete with probability one. This property can be seen from the explicit stick-breaking construction due to Sethuraman [11] as follows.

Let $(V_i)_{i=1}^{\infty}$ and $(\eta_i)_{i=1}^{\infty}$ be independent sequences of i.i.d random variables: $V_i \sim \text{Beta}(1, \alpha)$ and $\eta_i \sim G_0$. Then a random measure $G$ defined as

$$\theta_i = V_i \prod_{j=1}^{i-1} (1 - V_j) \qquad\qquad G = \sum_{i=1}^{\infty} \theta_i \delta_{\eta_i} \qquad (2)$$

where $\delta_\eta$ is a probability measure concentrated at $\eta$, is a random probability measure distributed according to $\text{DP}(\alpha, G_0)$ as shown in [11] .

## 3.2 Dirichlet Process Mixture Model (DPMM)

Dirichlet process has been used as a nonparametric prior on the parameters of a mixture model. This model is referred to as Dirichlet Process Mixture Model. Let $z$ be the mixture membership indicator variables for data variables $\mathbf{x}$. Using the stick-breaking construction in (2), the Dirichlet process mixture model is given by

$$G \sim \text{DP}(\alpha, G_0) \qquad z_i, \eta_i \mid G \sim G \qquad x_i \mid z_i, \eta_i \sim F(\eta_i) \qquad (3)$$

where $F(\eta_i)$ denotes the distribution of the observation $x_i$ given parameter $\eta_i$.

## 3.3 Dirichlet Process Mixture Model with Latent Matchings (DPMMLM)

In this section, we describe the new mixture model based on DP with latent variables for data matching between $\mathbf{x}$ and $\mathbf{y}$. We use $F_X(\eta), F_Y(\eta)$ to denote the marginal distribution of $X$ and $Y$ respectively; and $F_{X|Y}(y, \eta)$ to denote the conditional distribution of $X$ given $Y$. The parameter $\eta$ is a random variable of the prior distribution $G_0(\eta \mid \lambda_0)$ with hyperparameter $\lambda_0$. Also, let $z_i$ be the mixture membership of the sample pair $(x_i, y_{m_i})$. Our model is given by:

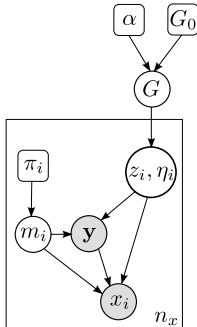

$$
\begin{aligned}
G &\sim \text{DP}(\alpha, G_0) \\
z_i, \eta_i \mid G &\sim G \\
m_i \mid \boldsymbol{\pi_i} &\sim \text{Discrete}(\boldsymbol{\pi_i}) \\
y_{m_i} \mid m_i, z_i, \eta_i &\sim F_Y(\eta_i), \text{if } m_i > 0 \\
x_i \mid m_i, z_i, \eta_i, \mathbf{y} &\sim \begin{cases} F_{X|Y}(y_{m_i}, \eta_i) & \text{if } m_i > 0 \\ F_X(\eta_i) & \text{otherwise} \end{cases}
\end{aligned} \qquad (4)
$$

The major difference between our model and a regular DPMM is the dependence of $x_i$ on $y$ if

$m_i > 0$. In other words the assignment of $x$ to a cluster depends on both, its own expression levels and the levels of the $y$ component to which it is matched. If $x$ is not matched to any $y$ component then we resort to the marginal distribution $F_X$ of the mixture.

## 3.4 Mean-field variational methods

For probabilistic models, mean-field variational methods [12, 13] provide a deterministic and bounded approximation to the intractable joint probability of observed and hidden variables. Briefly, given a model with observed variables $x$ and hidden variables $h$, we would like to compute $\log p(x)$, which requires us to marginalize over all hidden variables $h$. Since $p(x, h)$ is often intractable, we can find a tractable probability $q(h)$ that gives the best lower bound of $\log p(x)$ using Jensen 's inequality:

$$\log p(x) \geq \int_h q(h) \log p(x, h) - q(h) \log q(h) \ dh = \mathrm{E}_q[\log p(x, h)] - \mathrm{E}_q[\log q(h)] \quad (5)$$

Maximizing this lower bound is equivalent to finding the distribution $q(h)$ that minimizes the KL divergence between $q(h)$ and $p(h \mid x)$. Hence, $q(h)$ is the best approximation model within the chosen parametric family.

## 3.5 Variational Inference for DPMMLM

Although the DP mixture model is an "infinite" mixture model, it is intractable to solve the optimization problem when allowing for infinitely many variables. We thus follow the truncation approach used in [14], and limit the number of cluster to $K$. When $K$ is chosen to be large enough, the distribution is a drawn from the Dirichlet process [14]. To restrict the number of clusters to $K$, we set $V_K = 1$ and thus obtain $\theta_{i>K} = 0$ in (2). The likelihood of the observed data is

$$p(\mathbf{x}, \mathbf{y} \mid \alpha, \lambda_0) = \int_{\mathbf{m}, \mathbf{z}, \mathbf{v}, \boldsymbol{\eta}} p(\boldsymbol{\eta} \mid \lambda_0) \, p(\mathbf{v} \mid \alpha) \prod_{i=1}^{n_x} p(z_i \mid \mathbf{v})$$

$$\prod_{k=1}^{K} \left\{ \left( \pi_{i,0} f_X(x_i \mid \eta_k) \right)^{m_i^0} \prod_{j=1}^{n_y} \left( \pi_{i,j} f_{X|Y}(x_i \mid y_j, \eta_k) f_Y(y_j \mid \eta_k) \right)^{m_i^j} \right\}^{z_i^k} \quad (6)$$

where $p(z_i \mid \mathbf{v}) = v_{z_i} \prod_{k=1}^{z_i - 1}(1 - v_k)$ and $\mathbf{v}$ is the stick breaking variables given in Section 3.1. The first part of (6) $p(\boldsymbol{\eta} \mid \lambda_0) \, p(\mathbf{v} \mid \alpha)$ is the likelihood of the model parameters and the second part is the likelihood of the assignments to clusters and matchings.

Following the variational inference framework for conjugate-exponential graphical models [15] we choose the distribution that factorizes over $\{m_i, z_i\}_{i=1,\dots,n_x}$, $\{v_k\}_{k=1,\dots,K}$ and $\{\eta_k\}_{k=1,\dots,K-1}$ as follows:

$$q(\mathbf{m}, \mathbf{z}, \mathbf{v}, \boldsymbol{\eta}) = \prod_{i=1}^{n_x} \left\{ q_{\phi_i}(m_i) \prod_{j=0}^{n_y} q_{\theta_{i,j}}(z_i)^{m_i^j} \right\} \prod_{k=1}^{K-1} q_{\gamma_k}(v_k) \prod_{k=1}^{K} q_{\lambda_k}(\eta_k) \quad (7)$$

where $q_{\phi_i}(m_i)$ and $q_{\theta_{i,j}}(z_i)$ are multinomial distributions and $q_{\gamma_k}(v_k)$ are beta distributions. These distributions are conjugate distributions for the likelihood of the parameters in (6). $q_{\lambda_k}(\eta_k)$ requires special treatment due to the coupling of the marginal and conditional distributions in the likelihood. These issues are discussed in details in section 3.5.2.

Using this variational distribution we obtain a lower bound for the log likelihood:

$$\log p(\mathbf{x}, \mathbf{y} \mid \alpha, \lambda_0) \geq \mathrm{E}[\log p(\boldsymbol{\eta} \mid \lambda_0)] + \mathrm{E}[\log p(\mathbf{V} \mid \alpha)]$$

$$+ \sum_{i=1}^{n_x} \left\{ \mathrm{E}[\log p(Z_i \mid \mathbf{V})] + \sum_{j=0}^{n_y} \sum_{k=1}^{K} \mathrm{E}[M_i^j Z_i^k](\log \pi_{i,j} + \rho_{i,j,k}) \right\} - \mathrm{E}[\log q(\mathbf{M}, \mathbf{Z}, \mathbf{V}, \boldsymbol{\eta})] \quad (8)$$

where all expectations are with respect to the distribution $q(\mathbf{m}, \mathbf{z}, \mathbf{v}, \boldsymbol{\eta})$ and

$$\rho_{i,j,k} = \begin{cases} \mathrm{E}[\log f_{X|Y}(X_i \mid Y_j, \eta_k)] + \mathrm{E}[\log f_Y(Y_j \mid \eta_k)] & \text{if } j > 0 \\ \mathrm{E}[\log f_X(X_i \mid \eta_k)] & \text{if } j = 0 \end{cases}$$

To compute the terms in (8), we note that

$$\mathrm{E}[M_i^j Z_i^k] = \phi_{i,j}\theta_{i,j,k} = \psi_{i,j,k}$$

$$\mathrm{E}[\log p(Z_i \mid \mathbf{V})] = \sum_{k=1}^K q(z_i > k)\mathrm{E}[\log(1 - V_k)] + q(z_i = k)\mathrm{E}[\log V_k]$$

where $q(z_i > k) = \sum_{j=0}^{n_y}\sum_{t=k+1}^K \psi_{i,j,t}$ and $q(z_i = k) = \sum_{j=0}^{n_y} \psi_{i,j,k}$.

### 3.5.1 Coordinate ascent inference algorithm

The lower bound above can be optimized by a coordinate ascent algorithm. The update rules for all terms except for the $q_{\boldsymbol{\lambda}}(\boldsymbol{\eta})$, are presented below. These are direct applications of the variational inference for conjugate-exponential graphical models [15]. We discuss the update rule for $q_{\boldsymbol{\lambda}}(\boldsymbol{\eta})$ in section 3.5.2.

- Update for $q_{\gamma_k}(v_k)$:

$$\gamma_{k,1} = 1 + \sum_{i=1}^{n_x}\sum_{j=0}^{n_y}\psi_{i,j,k} \qquad \gamma_{k,2} = \alpha + \sum_{i=1}^{n_x}\sum_{j=0}^{n_y}\sum_{t=k+1}^K \psi_{i,j,t}$$

- Update for $q_{\theta_{i,j}}(z_i)$ and $q_{\phi_i}(m_i)$:

$$\theta_{i,j,k} \;\propto\; \exp\left(\rho_{i,j,k} + \sum_{k=1}^{k-1}\mathrm{E}[\log(1 - V_k)] + \mathrm{E}[\log V_k]\right)$$

$$\phi_{i,j} \;\propto\; \exp\left(\log\pi_{i,j} + \sum_{k=1}^K \theta_{i,j,k}\left(\rho_{i,j,k} + \sum_{k=1}^{k-1}\mathrm{E}[\log(1 - V_k)] + \mathrm{E}[\log V_k]\right)\right)$$

### 3.5.2 Application of the model to multivariate Gaussians

The previous sections described the model in a general terms. In the rest of this section, and in our experiments, we focus on data that is assumed to be distributed as a multivariate Gaussian with unknown mean and covariance matrix. The prior distribution $G_0$ is then given by the conjugate prior Gaussian-Wishart distribution. In a classical DP Gaussian Mixture Model with Gaussian-Wishart prior, the posterior distribution of the parameters could be computed analytically. Unfortunately, in our model, the coupling of the conditional and marginal distribution in the likelihood makes it difficult to derive analytical formulas for the posterior distribution. Note that if $(X, Y) \sim \mathcal{N}(\mu, \Sigma)$ with $\mu = (\mu_X, \mu_Y)$ and $\Sigma = \begin{pmatrix} \Sigma_X & \Sigma_{XY} \\ \Sigma_{YX} & \Sigma_Y \end{pmatrix}$ then $X \sim \mathcal{N}(\mu_X, \Sigma_X), Y \sim \mathcal{N}(\mu_Y, \Sigma_Y)$ and

$$X|Y = y \sim \mathcal{N}(\mu_X + \Sigma_{XY}\Sigma_Y^{-1}(y - \mu_Y), \Sigma_X - \Sigma_{XY}\Sigma_Y^{-1}\Sigma_{YX}). \qquad (9)$$

Therefore, we introduce an approximation distribution for the datasets which decouples the marginal and conditional distributions as follows:

$$f_X(x \mid \mu_X, \Lambda_X) \;=\; \mathcal{N}(\mu_X, \Sigma = \Lambda_X^{-1}) \qquad f_Y(y \mid \mu_Y, \Lambda_Y) \;=\; \mathcal{N}(\mu_Y, \Sigma = \Lambda_Y^{-1})$$

$$f_{X|Y}(x \mid y, \mathbf{W}, b, \mu_X, \Lambda_X) \;=\; \mathcal{N}(\mu_X + b - \mathbf{W}y, \Sigma = \Lambda_X^{-1})$$

where $\mathbf{W}$ is a $p_x \times p_y$ projection matrix and $\Lambda$ is the precision matrix. In this approximation, we assume that the covariance matrices of $X$ and $X|Y$ are the same. In other words, the covariance of $X$ is independent of $Y$. The matrix $\mathbf{W}$ models the linear correlation of $X$ on $Y$, similar to $-\Sigma_{XY}\Sigma_Y^{-1}$ in (9).

The priors for $\mu_X, \Lambda_X$ and $\mu_Y, \Lambda_Y$ are given by Gaussian-Wishart(GW) distributions. A flat improper prior is given to $\mathbf{W}$ and $b$, $p_0(\mathbf{W}) = 1, p_0(b) = 1$ for all $\mathbf{W}, b$. These assumptions lead to decoupling of the marginal and conditional distributions. Therefore, the distribution $q_{\lambda_k}(\eta_k)$ can now be factorized into two GW distributions and a distribution of $\mathbf{W}$. To avoid over-cluttering symbols, we omit the subscript $k$ of the specific cluster $k$.

$$q_{\lambda_k}^*(\eta_k) = GW(\mu_X, \Lambda_X)\, GW(\mu_Y, \Lambda_Y)\, g(\mathbf{W})\, g(b)$$

*Posterior distribution of $\mu_Y, \Lambda_Y$:* The update rules follow the standard posterior distribution of Gaussian-Wishart conjugate priors.

*Posterior distribution of $\mu_X, \Lambda_X$ and $\mathbf{W}, b$:* Due to the coupling of $\mu_X, \Lambda_X$ with $\mathbf{W}$, we do a coordinate ascent procedure to find the optimal posterior distribution. The posterior distribution of $\mathbf{W}, b$ is a singleton discrete distribution $g$ such that $g(\mathbf{W}^*) = 1, g(b^*) = 1$.

- Update for posterior distribution of $\mu_X, \Lambda_X$:

$$\kappa_X = \kappa_{X0} + n_X \qquad\qquad\qquad m_X = \frac{1}{\kappa_X}(\kappa_{X0}m_{X0} + n_X\overline{x})$$

$$S_X^{-1} = S_{X0}^{-1} + V_X + \frac{\kappa_{X0}n_X}{\kappa_{X0} + n_X}(\overline{x} - m_{X0})(\overline{x} - m_{X0})^T \qquad \nu_X = \nu_{X0} + n_X$$

where $n_X = \sum_{n=1}^{n_x}\sum_{j=0}^{n_y}\psi_{i,j,k}$, $\overline{x} = \frac{1}{n_X}\sum_{i=1}^{n_x}\left(\psi_{i,0,k}x_i + \sum_{j=1}^{n_y}\psi_{i,j,k}(x_i - b + \mathbf{W}^*y_j))\right)$ and

$$V_X = \sum_{i=1}^{n_x}\left\{\psi_{i,0,k}(x_i - \overline{x})(x_i - \overline{x})^T + \sum_{j=1}^{n_y}\psi_{i,j,k}(x_i - b + \mathbf{W}^*y_j - \overline{x})(x_i - b + \mathbf{W}^*y_j - \overline{x})^T\right\}.$$

- Update for $\mathbf{W}^*, b^*$: We find $\mathbf{W}^*, b^*$ that maximizes the log likelihood. Taking the derivative with respect to $\mathbf{W}^*$ and solving for $\mathbf{W}^*$, we get

$$\mathbf{W}^* = \left(\sum_{i=1}^{n_x}\sum_{j=1}^{n_y}\psi_{i,j,k}(x_i - m_X - b)y_j^T\right)\left(\sum_{i=1}^{n_x}\sum_{j=1}^{n_y}\psi_{i,j,k}y_jy_j^T\right)^{-1}$$

$$b^* = -\left(\sum_{i=1}^{n_x}\sum_{j=1}^{n_y}\psi_{i,j,k}(x_i - m_X + \mathbf{W}^*y_j)\right)/\sum_{i=1}^{n_x}\sum_{j=1}^{n_y}\psi_{i,j,k}$$

## 4   Experiments and Results

### 4.1   Simulated data

We demonstrate the performance of the model in identifying data matchings as well as cluster membership of datapoints using simulated data. To generate a simulated dataset, we sample 120 datapoints from a mixture of three 5-dimensional Gaussians with separation coefficient = 2 leading to well separated mixtures[1]. The covariance matrix was derived from the autocorrelation matrix for a first-order autoregressive process leading to highly dependent components ($\rho = 0.9$). From these samples, we use the first 3 dimensions to create 120 datapoints $\mathbf{x} = [x_1, \ldots, x_{120}]$. The last two dimensions of the first 100 datapoints are used to create $\mathbf{y} = [y_1, \ldots, y_{100}]$ (note that there are no matches for 20 points in $x$). Hence, the ground truth $\mathcal{M}$ matrix is a diagonal $120 \times 100$ matrix. We selected a large value for the diagonal entries ($\tau = 1000$) in order to place a strong prior for the correct matchings. Next, for $t = 0, \ldots, 20$, we randomly select $t$ entries on each row of $\mathcal{M}$ and set them to $\frac{\tau}{2}r$, where $r \sim \chi_1^2$. We repeat the process 20 times for each $t$ to compute the mean and standard deviation shown in Figure 1(a) and Figure 1(b). We compare the performance of our model(DPMMLM) with a standard Dirichlet Process Mixture Model where each component in $\mathbf{x}$ is matched based on the highest prior: $\{(x_i, y_{j^*}) \mid i = 1, \ldots, 100 \text{ and } j^* = \text{argmax}_j \mathcal{M}(i, j)\}$ (DPMM). For all models, the truncation level ($K$) is set to 20 and $\alpha$ is 1. Figure 1(a) presents the percentage of correct matchings inferred by DPMMLM and the highest prior matching. For DPMMLM, a datapoint $x_i$ is matched to the datapoint $y_j$ with the largest posterior probability $\phi_{i,j}$. With the added noise, DPMMLM can still achieve an accuracy of 50% when the highest prior matching leads to only 25% accuracy. Figure 1(b) and 1(c) show the Normalized Mutual Information (NMI) and Adjusted Rand index [17] for the clusters inferred by the two models compared to the true clusters. As can be seen, while the percentage of correct matchings decreased with the added noise, DPMMLM still achieves high NMI of 0.8 and Adjusted Rand index of 0.92. In conclusion, by relying on matchings of points DPMMLM can still performs very well in terms of its ability to identify correct clusters even with the high noise levels.

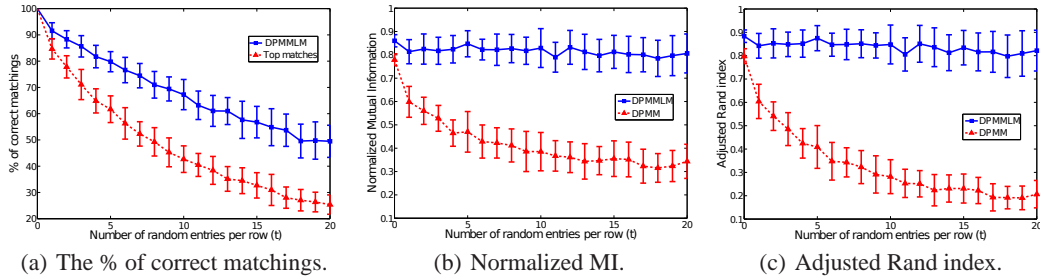

(a) The % of correct matchings.     (b) Normalized MI.     (c) Adjusted Rand index.

Figure 1: Evaluation of the result on simulated data.

## 4.2 Immune response dataset

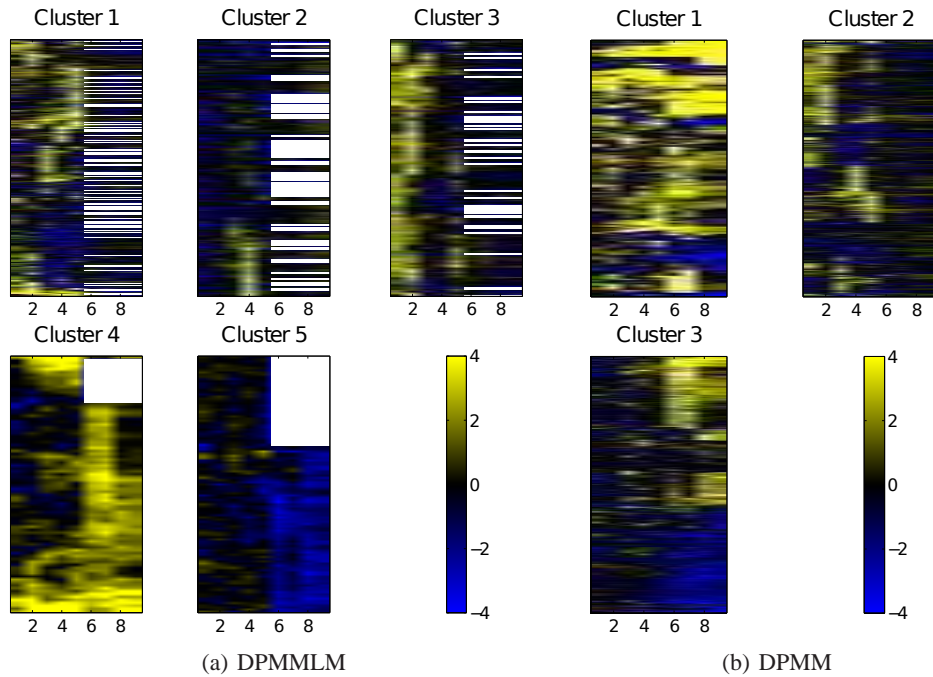

(a) DPMMLM          (b) DPMM

Figure 2: The heatmap for clusters inferred for the immune response dataset.

We compared human and mouse immune response datasets to identify similar and divergent genes. We selected two experiments that studied immune response to gram negative bacteria. The first was a time series of human response to Salmonella [18]. Cells were infected with Salmonella and were profiled at: 0.5h, 1h, 2h, 3h and 4h. The second looked at mouse response to Yersinia enterocolitica with and without treatment by IFN-$\gamma$ [19]. We used BLASTN to compute the sequence similarity (bit-score) between all human and mouse genes. For each species we selected the most varying 500 genes and expanded the gene list to include all matched genes in the other species with a bit score greater than 75. This led to a set of 1476 human and 1967 mouse genes which we compared using our model. The $\mathcal{M}$ matrix is the bit scores between human and mouse genes thresholded at 75.

The resulting clusters are presented in Figure 2(a). In that figure, the first five dimensions are human expression values and each gene in human is matched to the mouse gene with the highest posterior. Human genes which are not matched to any mouse gene in the cluster have a blank line on the mouse side of the figure. The algorithm identified five different clusters. Clusters 1, 4 and 5 display a similar expression pattern in human and mouse with genes either up or down regulated in response to the infection. Genes in cluster 2 differ between the two species being mostly down regulated in humans while slightly upregulated in mouse. Human genes in cluster 3 also differ from their mouse orthologs. While they are strongly upregulated in humans, the corresponding mouse genes do not change much.

| P value | Corrected P | GO term description | | P value | Corrected P | GO term description |
|---|---|---|---|---|---|---|
| 2.86216e-10 | <0.001 | regulation of apoptosis | | 5.06685e-07 | 0.001 | response to stimulus |
| 4.97408e-10 | <0.001 | regulation of cell death | | 6.15795e-07 | 0.001 | negative regulation of biological process |
| 7.82427e-10 | <0.001 | protein binding | | 7.70651e-07 | 0.001 | cellular process |
| 4.14320e-10 | <0.001 | regulation of programmed cell death | | 7.78266e-07 | 0.002 | regulation of localization |
| 4.49332e-09 | <0.001 | positive regulation of cellular process | | 1.09778e-06 | 0.002 | response to organic substance |
| 4.77653e-09 | <0.001 | positive regulation of biological process | | 1.42704e-06 | 0.002 | collagen metabolic process |
| 8.27313e-09 | <0.001 | response to chemical stimulus | | 1.91735e-06 | 0.003 | negative regulation of cellular process |
| 1.17013e-07 | 0.001 | cytoplasm | | 3.23244e-06 | 0.005 | multicellular organismal macromolecule metabolic process |
| 1.28299e-07 | 0.001 | response to stress | | 3.39901e-06 | 0.005 | interspecies interaction |
| 2.20104e-07 | 0.001 | cell proliferation | | 3.66178e-06 | 0.005 | negative regulation of apoptosis |

Table 1: The GO enrichment result for cluster 1 identified by DPMMLM.

We used the Gene Ontology (GO, www.geneontology.org) to calculate the enrichment of functional categories in each cluster based on the hypergeometric distribution. Genes in cluster 1 (Table 1) are associated with immune and stress responses. Interestingly the most significant category for this cluster is "regulation of apoptosis" (corrected p-value <0.001). Indeed, both Salmonella and Yersinia are known to induce apoptosis in host cells [20]. When clustering the two datasets independently the p-value for this category is greatly reduced indicating that accurate matchings can lead to better identification of core pathways (see Appendix). Cluster 4 contains the most coherent set of upregulated genes across the two species. One of top GO categories for this cluster is 'response to molecule of bacterial origin' (corrected p-value < 0.001) which is the most accurate description of the condition tested. See Appendix for complete GO tables of all clusters. In contrast to clusters in which mouse and human genes are similarly expressed, cluster 3 genes are strongly upregulated in human cells while not changing in mouse. This cluster is enriched for ribosomal proteins (corrected p-value <0.001). This may indicate different strategies utilized by the bacteria in the two experiments. There are studies that show that pathogens can upregulate the synthesis of ribosomal genes (which are required for translation) [21] whereas other studies indicate that ribosomal genes may not change much, or may even be reduced, following infection [22]. The results of our analysis indicate that while following Salmonella infection in human cells ribosomal genes are upregulated, they are not activated following Yarsinia infection in mouse.

We have also analyzed the matchings obtained using sequence data alone (prior) and by combining sequence and expression data (posterior) using our method. The top posterior gene is the same as the top prior gene in most cases (905 of the 1476 human genes). However, there are several cases in which the prior and posterior differ. 293 human genes are not matched to any mouse gene in the cluster they are assigned to indicating that they are expressed in a species dependent manner. Additionally, for 278 human genes the top posterior and prior mouse gene differ. To test whether these differences inferred by the algorithm are biologically meaningful we compared our Dirichlet method to a method that uses deterministic assignments, as was done in the past. Using such assignments the algorithm identified only three clusters as shown in Figure 2(b). Neither of these clusters looked homogenous across species.

## 5  Conclusions

We have developed a new model for simultaneously clustering and matching genes across species. The model uses a Dirichlet Process to infer the number of clusters. We developed an efficient variational inference method that scales to large datasets with almost 2000 datapoints. We have also demonstrated the power of our method on simulated data and immune response dataset. While the method was presented in the context of expression data it is general and can be used for other matching tasks in which a prior can be obtained. For example, when trying to determine a caption for images extracted from webpages a prior can be obtained by relying on the distance between the image and the text on the page. Next, clustering can be employed to utilize the abundance of images that are extracted and improve the matching outcome.

**Acknowledgments**

We thank the anonymous reviewers for constructive and insightful comments. This work is supported in part by NIH grant 1RO1 GM085022 and NSF grants DBI-0965316 and CAREER-0448453 to Z.B.J.

## Footnotes

[1] Following [16], a Gaussian mixture is c-separated if for each pair $(i, j)$ of components, $\|m_i - m_j\|^2 \geq c^2 D \max(\lambda_i^{\max}, \lambda_j^{\max})$, where $\lambda^{\max}$ denotes the maximum eigenvalue of their covariance.

# References

[1] M. Kellis, N. Patterson, M. Endrizzi, B. Birren, and E. S. Lander. Sequencing and comparison of yeast species to identify genes and regulatory elements. *Nature*, 423:241–254, May 2003.

[2] L. J. Jensen, T. S. Jensen, U. de Lichtenberg, S. Brunak, and P. Bork. Co-evolution of transcriptional and post-translational cell-cycle regulation. *Nature*, 443:594–597, Oct 2006.

[3] G. Lelandais et al. Genome adaptation to chemical stress: clues from comparative transcriptomics in Saccharomyces cerevisiae and Candida glabrata. *Genome Biol.*, 9:R164, 2008.

[4] G. Quon, Y. W. Teh, E. Chan, M. Brudno, T. Hughes, and Q. D. Morris. A mixture model for the evolution of gene expression in non-homogeneous datasets. In *Advances in Neural Information Processing Systems*, volume 21, 2009.

[5] J. M. Stuart, E. Segal, D. Koller, and S. K. Kim. A gene-coexpression network for global discovery of conserved genetic modules. *Science*, 302:249–255, Oct 2003.

[6] Sven Bergmann, Jan Ihmels, and Naama Barkai. Similarities and differences in genome-wide expression data of six organisms. *PLoS Biol*, 2(1):e9, 12 2003.

[7] Y. Lu, R. Rosenfeld, and Z. Bar-Joseph. Identifying cycling genes by combining sequence homology and expression data. *Bioinformatics*, 22:e314–322, Jul 2006.

[8] Y. Lu, R. Rosenfeld, G. J. Nau, and Z. Bar-Joseph. Cross species expression analysis of innate immune response. *J. Comput. Biol.*, 17:253–268, Mar 2010.

[9] R. Sharan et al. Conserved patterns of protein interaction in multiple species. *Proc. Natl. Acad. Sci. U.S.A.*, 102:1974–1979, Feb 2005.

[10] Thomas S. Ferguson. A bayesian analysis of some nonparametric problems. *The Annals of Statistics*, 1(2):209–230, 1973.

[11] J. Sethuraman. A constructive definition of dirichlet priors. *Statistica Sinica*, 4:639–650, 1994.

[12] Michael I. Jordan, Zoubin Ghahramani, Tommi S. Jaakkola, and Lawrence K. Saul. An introduction to variational methods for graphical models. *Machine Learning*, 37(2):183–233, November 1999.

[13] Martin J. Wainwright and Michael I. Jordan. Graphical models, exponential families, and variational inference. *Found. Trends Mach. Learn.*, 1(1-2):1–305, 2008.

[14] H. Ishwaran and James. Gibbs sampling methods for stick breaking priors. *Journal of the American Statistical Association*, pages 161–173, March 2001.

[15] Zoubin Ghahramani and Matthew J. Beal. Propagation algorithms for variational bayesian learning. In *In Advances in Neural Information Processing Systems 13*, pages 507–513. MIT Press, 2001.

[16] Sanjoy Dasgupta. Learning mixtures of gaussians. In *FOCS '99: Proceedings of the 40th Annual Symposium on Foundations of Computer Science*, Washington, DC, USA, 1999.

[17] M. Meila. Comparing clusterings by the variation of information. In *Learning theory and Kernel machines: 16th Annual Conference on Learning Theory and 7th Kernel Workshop, COLT/Kernel 2003, Washington, DC, USA, August 24-27, 2003: proceedings*, page 173. Springer Verlag, 2003.

[18] C. S. Detweiler et al. Host microarray analysis reveals a role for the Salmonella response regulator phoP in human macrophage cell death. *Proc. Natl. Acad. Sci. U.S.A.*, 98:5850–5855, May 2001.

[19] K. van Erp et al. Role of strain differences on host resistance and the transcriptional response of macrophages to infection with Yersinia enterocolitica. *Physiol. Genomics*, 25:75–84, 2006.

[20] D. M. Monack, B. Raupach, et al. Salmonella typhimurium invasion induces apoptosis in infected macrophages. *Proc. Natl. Acad. Sci. U.S.A.*, 93:9833–9838, Sep 1996.

[21] O. O. Zharskaia et al. [Activation of transcription of ribosome genes following human embryo fibroblast infection with cytomegalovirus in vitro]. *Tsitologiia*, 45:690–701, 2003.

[22] J. W. Gow, S. Hagan, P. Herzyk, C. Cannon, P. O. Behan, and A. Chaudhuri. A gene signature for post-infectious chronic fatigue syndrome. *BMC Med Genomics*, 2:38, 2009.

